# Fast Pruning Using Principal Components

**Asriel U. Levin, Todd K. Leen and John E. Moody**
Department of Computer Science and Engineering
Oregon Graduate Institute
P.O. Box 91000
Portland, OR 97291-1000

## Abstract

We present a new algorithm for eliminating excess parameters and improving network generalization after supervised training. The method, "Principal Components Pruning (PCP)", is based on principal component analysis of the node activations of successive layers of the network. It is simple, cheap to implement, and effective. It requires no network retraining, and does not involve calculating the full Hessian of the cost function. Only the weight and the node activity correlation matrices for each layer of nodes are required. We demonstrate the efficacy of the method on a regression problem using polynomial basis functions, and on an economic time series prediction problem using a two-layer, feedforward network.

## 1 Introduction

In supervised learning, a network is presented with a set of training exemplars $[u(k), y(k)]$, $k = 1 \ldots N$ where $u(k)$ is the $k^{th}$ input and $y(k)$ is the corresponding output. The assumption is that there exists an underlying (possibly noisy) functional relationship relating the outputs to the inputs

$$y = f(u, e)$$

where $e$ denotes the noise. The aim of the learning process is to approximate this relationship based on the the training set. The success of the learned approximation

is judged by the ability of the network to approximate the outputs corresponding to inputs it was not trained on.

Large networks have more functional flexibility than small networks, so are better able to fit the training data. However large networks can have higher parameter variance than small networks, resulting in poor generalization. The number of parameters in a network is a crucial factor in it's ability to generalize.

No practical method exists for determining, a priori, the proper network size and connectivity. A promising approach is to start with a large, fully-connected network and through pruning or regularization, increase model bias in order to reduce model variance and improve generalization.

**Review of existing algorithms**

In recent years, several methods have been proposed. Skeletonization (Mozer and Smolensky, 1989) removes the neurons that have the least effect on the output error. This is costly and does not take into account correlations between the neuron activities. Eliminating small weights does not properly account for a weight's effect on the output error. *Optimal Brain Damage (OBD)* (Le Cun et al., 1990) removes those weights that least affect the training error based on a diagonal approximation of the Hessian. The diagonal assumption is inaccurate and can lead to the removal of the wrong weights. The method also requires retraining the pruned network, which is computationally expensive. *Optimal Brain Surgeon (OBS)* (Hassibi et al., 1992) removes the "diagonal" assumption but is impractical for large nets. Early stopping monitors the error on a validation set and halts learning when this error starts to increase. There is no guarantee that the learning curve passes through the optimal point, and the final weight is sensitive to the learning dynamics. Weight decay (ridge regression) adds a term to the objective function that penalizes large weights. The proper coefficient for this term is not known a priori, so one must perform several optimizations with different values, a cumbersome process.

We propose a new method for eliminating excess parameters and improving network generalization. The method, "Principal Components Pruning (PCP)", is based on principal component analysis (PCA) and is simple, cheap and effective.

## 2   Background and Motivation

PCA (Jolliffe, 1986) is a basic tool to reduce dimension by eliminating redundant variables. In this procedure one transforms variables to a basis in which the covariance is diagonal and then projects out the low variance directions.

While application of PCA to remove input variables is useful in some cases (Leen et al., 1990), there is no guarantee that low variance variables have little effect on *error*. We propose a saliency measure, based on PCA, that identifies those variables that have the least effect on error. Our proposed Principal Components Pruning algorithm applies this measure to obtain a simple and cheap pruning technique in the context of supervised learning.

**Special Case: PCP in Linear Regression**

In unbiased linear models, one can bound the bias introduced from pruning the principal degrees of freedom in the model. We assume that the observed system is described by a signal-plus-noise model with the signal generated by a function linear in the weights:

$$y = W_0 u + e$$

where $u \in \Re^p$, $y \in \Re^m$, $W \in \Re^{m \times p}$, and $e$ is a zero mean additive noise. The regression model is

$$\hat{y} = W u .$$

The input correlation matrix is $\Sigma = \frac{1}{N} \sum_k u(k) u^T(k)$.

It is convenient to define coordinates in which $\Sigma$ is diagonal $\Lambda \equiv C^T \Sigma C$ where $C$ is the matrix whose columns are the orthonormal eigenvectors of $\Sigma$. The transformed input variables and weights are $\tilde{u} = C^T u$ and $\tilde{W} = W C$ respectively, and the model output can be rewritten as $\hat{y} = \tilde{W} \tilde{u}$ .

It is straightforward to bound the increase in training set error resulting from removing subsets of the transformed input variable. The sum squared error is

$$I \equiv \frac{1}{N} \sum_k [y(k) - \hat{y}(k)]^T [y(k) - \hat{y}(k)]$$

Let $\hat{y}_l(k)$ denote the model's output when the last $p - l$ components of $\tilde{u}(k)$ are set to zero. By the triangle inequality

$$\begin{aligned} I_l &\equiv \frac{1}{N} \sum_k [y(k) - \hat{y}_l(k)]^T [y(k) - \hat{y}_l(k)] \\ &\leq I + \frac{1}{N} \sum_k [\hat{y}(k) - \hat{y}_l(k)]^T [\hat{y}(k) - \hat{y}_l(k)] . \end{aligned} \tag{1}$$

The second term in (1) bounds the increase in the training set error[1]. This term can be rewritten as

$$\frac{1}{N} \sum_k [\hat{y}(k) - \hat{y}_l(k)]^T [\hat{y}(k) - \hat{y}_l(k)] = \sum_{i=l+1}^p \tilde{w}_i^T \tilde{w}_i \lambda_i$$

where $\tilde{w}_i$ denotes the $i^{th}$ column of $\tilde{W}$ and $\lambda_i$ is the $i^{th}$ eigenvalue. The quantity $\tilde{w}_i^T \tilde{w}_i \lambda_i$ measures the effect of the $i^{th}$ eigen-coordinate on the output error; it serves as our saliency measure for the weight $\tilde{w}_i$.

Relying on Akaike's Final Prediction error (FPE) (Akaike, 1970), the average *test set* error for the original model is given by

$$J[W] = \frac{N + pm}{N - pm} I(W)$$

where $pm$ is the number of parameters in the model. If $p - l$ principal components are removed, then the expected test set is given by

$$J_l[W] = \frac{N + lm}{N - lm} I_l(W) .$$

[1] For $y \in R^1$, the inequality is replaced by an equality.

If we assume that $N \gg l * m$, the last equation implies that the optimal generalization will be achieved if all principal components for which

$$\tilde{w}_i^T \tilde{w}_i \lambda_i < \frac{2mI}{N}$$

are removed. For these eigen-coordinates the reduction in model variance will more then compensate for the increase in training error, leaving a lower expected test set error.

## 3    Proposed algorithm

The pruning algorithm for linear regression described in the previous section can be extended to multilayer neural networks. A complete analysis of the effects on generalization performance of removing eigen-nodes in a nonlinear network is beyond the scope of this short paper. However, it can be shown that removing eigen-nodes with low saliency reduces the *effective* number of parameters (Moody, 1992) and should usually improve generalization. Also, as will be discussed in the next section, our PCP algorithm is related to the OBD and OBS pruning methods. As with all pruning techniques and analyses of generalization, one must assume that the data are drawn from a stationary distribution, so that the training set fairly represents the distribution of data one can expect in the future.

Consider now a feedforward neural network, where each layer is of the form

$$y^i = \Gamma[W^i u^i] \equiv \Gamma[x^i] \ .$$

Here, $u^i$ is the input, $x^i$ is the weighted sum of the input, $\Gamma$ is a diagonal operator consisting of the activation function of the neurons at the layer, and $y^i$ is the output of the layer.

1. A network is trained using a supervised (e.g. backpropagation) training procedure.

2. Starting at the first layer, the correlation matrix $\Sigma$ for the input vector to the layer is calculated.

3. Principal components are ranked by their effect on the linear output of the layer. [2]

4. The effect of removing an eigennode is evaluated using a validation set. Those that do not increase the validation error are deleted.

5. The weights of the layer are projected onto the $l$ dimensional subspace spanned by the significant eigenvectors

$$W \rightarrow W C_l C_l^T$$

   where the columns of $C$ are the eigenvectors of the correlation matrix.

6. The procedure continues until all layers are pruned.

As seen, the algorithm proposed is easy and fast to implement. The matrix dimensions are determined by the number of neurons in a layer and hence are manageable even for very large networks. No retraining is required after pruning and the speed of running the network after pruning is not affected.

**Note:** A finer scale approach to pruning should be used if there is a large variation between $\tilde{w}_{ij}$ for different $j$. In this case, rather than examine $\tilde{w}_i^T \tilde{w}_i \lambda_i$ in one piece, the contribution of each $\tilde{w}_{ij}^2 \lambda_i$ could be examined individually and those weights for which the contribution is small can be deleted.

## 4  Relation to Hessian-Based Methods

The effect of our PCP method is to reduce the rank of each layer of weights in a network by the removal of the least salient eigen-nodes, which reduces the *effective* number of parameters (Moody, 1992). This is in contrast to the OBD and OBS methods which reduce the rank by eliminating actual weights. PCP differs further from OBD and OBS in that it does not require that the network be trained to a local minimum of the error.

In spite of these basic differences, the PCP method can be viewed as intermediate between OBD and OBS in terms of how it approximates the Hessian of the error function. OBD uses a diagonal approximation, while OBS uses a linearized approximation of the full Hessian. In contrast, PCP effectively prunes based upon a block-diagonal approximation of the Hessian. A brief discussion follows.

In the special case of linear regression, the correlation matrix $\Sigma$ *is* the full Hessian of the squared error.[3] For a multilayer network with $Q$ layers, let us denote the numbers of units per layer as $\{p_q : q = 0 \ldots Q\}$.[4] The number of weights (including biases) in each layer is $b_q = p_q(p_{q-1} + 1)$, and the total number of weights in the network is $B = \sum_{q=1}^{Q} b_q$. The Hessian of the error function is a $B \times B$ matrix, while the input correlation matrix for each of the units in layer $q$ is a much simpler $(p_{q-1} + 1) \times (p_{q-1} + 1)$ matrix. Each layer has associated with it $p_q$ identical correlation matrices.

The combined set of these correlation matrices for all units in layers $q = 1 \ldots Q$ of the network serves as a linear, block-diagonal approximation to the full Hessian of the nonlinear network.[5] This block-diagonal approximation has $\sum_{q=1}^{Q} p_q(p_{q-1} + 1)^2$ non-zero elements, compared to the $[\sum_{q=1}^{Q} p_q(p_{q-1} + 1)]^2$ elements of the full Hessian (used by OBS) and the $\sum_{q=1}^{Q} p_q(p_{q-1} + 1)$ diagonal elements (used by OBD). Due to its greater richness in approximating the Hessian, we expect that PCP is likely to yield better generalization performance than OBD.

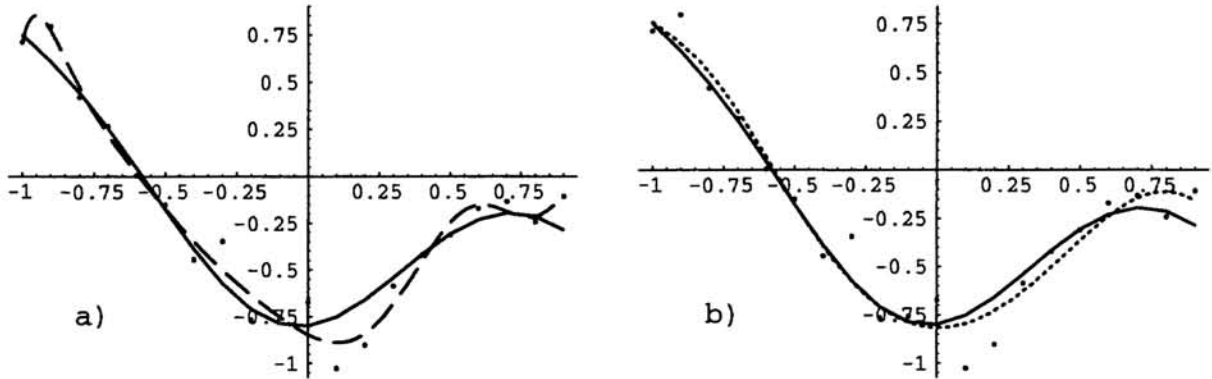

Figure 1: a) Underlying function (solid), training data (points), and $10^{th}$ order polynomial fit (dashed). b) Underlying function, training data, and pruned regression fit (dotted).

The computational complexities of the OBS, OBD, and PCP methods are

$$O\left(N\left[\sum_{q=1}^{Q}p_q(p_{q-1}+1)\right]^2\right)\ ,\ O\left(N\sum_{q=1}^{Q}p_q(p_{q-1}+1)\right)\ ,\ O\left(N\sum_{q=1}^{Q}(p_{q-1}+1)^2\right)$$

respectively, where we assume that $N \geq B$. The computational cost of PCP is therefore significantly less than that of OBS and is similar to that of OBD.

## 5   Simulation Results

**Regression With Polynomial Basis Functions**

The analysis in section 2 is directly applicable to regression using a linear combination of basis functions $\hat{y} = W f(u)$. One simply replaces $u$ with the vector of basis functions $f(u)$.

We exercised our pruning technique on a univariate regression problem using monomial basis functions $f(u) = (1, u, u^2, \ldots, u^n)^T$ with $n = 10$. The underlying function was a sum of four sigmoids. Training and test data were generated by evaluating the underlying function at 20 uniformly spaced points in the range $-1 \leq u \leq +1$ and adding gaussian noise. The underlying function, training data and the polynomial fit are shown in figure 1a.

The mean squared error on the training set was 0.00648. The test set mean squared error, averaged over 9 test sets, was 0.0183 for the unpruned model. We removed the eigenfunctions with the smallest saliencies $\tilde{w}^2 \lambda$. The lowest average test set error of 0.0126 was reached when the trailing four eigenfunctions were removed.[6] Figure 1b shows the *pruned* regression fit.

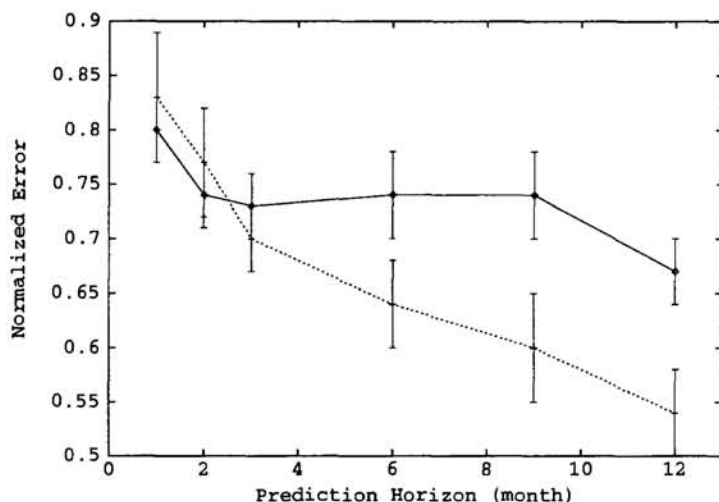

Figure 2: Prediction of the IP index 1980 - 1990. The solid line shows the performance before pruning and the dotted line the performance after the application of the PCP algorithm. The results shown represent averages over 11 runs with the error bars representing the standard deviation of the spread.

**Time Series Prediction with a Sigmoidal Network**

We have applied the proposed algorithm to the task of predicting the Index of Industrial Production (IP), which is one of the main gauges of U.S. economic activity. We predict the rate of change in IP over a set of future horizons based on lagged monthly observations of various macroeconomic and financial indicators (altogether 45 inputs). [7]

Our standard benchmark is the rate of change in IP for January 1980 to January 1990 for models trained on January 1960 to December 1979. In all runs, we used two layer networks with 10 tanh hidden nodes and 6 linear output nodes corresponding to the various prediction horizons (1, 2, 3, 6, 9, and 12 months). The networks were trained using stochastic backprop (which with this very noisy data set outperformed more sophisticated gradient descent techniques). The test set results with and without the PCP algorithm are shown in Figure 2.

Due to the significant noise and nonstationarity in the data, we found it beneficial to employ both weight decay and early stopping during training. In the above runs, the PCP algorithm was applied on top of these other regularization methods.

## 6   Conclusions and Extensions

Our "Principal Components Pruning (PCP)" algorithm is an efficient tool for reducing the *effective* number of parameters of a network. It is likely to be useful when there are correlations of signal activities. The method is substantially cheaper to implement than OBS and is likely to yield better network performance than OBD.[8]

Furthermore, PCP can be used on top of any other regularization method, including early stopping or weight decay.[9] Unlike OBD and OBS, PCP does not require that the network be trained to a local minimum.

We are currently exploring nonlinear extensions of our linearized approach. These involve computing a block-diagonal Hessian in which the block corresponding to each unit differs from the correlation matrix for that layer by a nonlinear factor. The analysis makes use of GPE (Moody, 1992) rather than FPE.

## Acknowledgements

One of us (TKL) thanks Andreas Weigend for stimulating discussions that provided some of the motivation for this work. AUL and JEM gratefully acknowledge the support of the Advanced Research Projects Agency and the Office of Naval Research under grant ONR N00014-92-J-4062. TKL acknowledges the support of the Electric Power Research Institute under grant RP8015-2 and the Air Force Office of Scientific Research under grant F49620-93-1-0253.

## Footnotes

[2] If we assume that $-\Gamma$ is the sigmoidal operator, relying on its contraction property, we have that the resulting output error is bounded by $\|e\| <= \|W\| \|e_{x1}\|$ where $e_{x1}$ is error observed at $x_i$ and $\|W\|$ is the norm of the matrices connecting it to the output.

[3]The correlation matrix and Hessian may differ by a numerical factor depending on the normalization of the squared error. If the error function is defined as one half the average squared error (ASE), then the equality holds.

[4]The inputs to the network constitute layer 0.

[5]The derivation of this approximation will be presented elsewhere. However, the correspondence can be understood in analogy with the special case of linear regression.

[6]The FPE criterion suggested pruning the trailing three eigenfunctions. We note that our example does not satisfy the assumption of an unbiased model, nor are the sample sizes large enough for the FPE to be completely reliable.

[7]Preliminary results on this problem have been described briefly in (Moody et al., 1993), and a detailed account of this work will be presented elsewhere.

[8]See section 4 for a discussion of the block-diagonal Hessian interpretation of our method. A systematic empirical comparison of computational cost and resulting network performance of PCP to other methods like OBD and OBS would be a worthwhile undertaking.

[9](Weigend and Rumelhart, 1991) called the rank of the covariance matrix of the node activities the "effective dimension of hidden units" and discussed it in the context of early stopping.

## References

Akaike, H. (1970). Statistical predictor identification. *Ann. Inst. Stat. Math.*, 22:203.

Hassibi, B., Stork, D., and Wolff, G. (1992). Optimal brain surgeon and general network pruning. Technical Report 9235, RICOH California Research Center, Menlo Park, CA.

Jolliffe, I. T. (1986). *Principal Component Analysis.* Springer-Verlag.

Le Cun, Y., Denker, J., and Solla, S. (1990). Optimal brain damage. In Touretzky, D., editor, *Advances in Neural Information Processing Systems*, volume 2, pages 598–605, Denver 1989. Morgan Kaufmann, San Mateo.

Leen, T. K., Rudnick, M., and Hammerstrom, D. (1990). Hebbian feature discovery improves classifier efficiency. In *Proceedings of the IEEE/INNS International Joint Conference on Neural Networks*, pages I–51 to I–56.

Moody, J. (1992). The effective number of parameters: An analysis of generalization and regularization in nonlinear learning systems. In Moody, J., Hanson, S., and Lippman, R., editors, *Advances in Neural Information Processing Systems*, volume 4, pages 847–854. Morgan Kaufmann.

Moody, J., Levin, A., and Rehfuss, S. (1993). Predicting the u.s. index of industrial production. *Neural Network World*, 3:791–794. in Proceedings of Parallel Applications in Statistics and Economics '93.

Mozer, M. and Smolensky, P. (1989). Skeletonization: A technique for trimming the fat from a network via relevance assesment. In Touretzky, D., editor, *Advances in Neural Information Processing Systems*, volume 1, pages 107–115. Morgan Kaufmann.

Weigend, A. S. and Rumelhart, D. E. (1991). Generalization through minimal networks with application to forecasting. In Keramidas, E. M., editor, *INTERFACE'91 – 23rd Symposium on the Interface: Computing Science and Statistics*, pages 362–370.

